# Obstacle Avoidance through Reinforcement Learning

**Tony J. Prescott** and **John E. W. Mayhew**
Artificial Intelligence and Vision Research Unit,
University of Sheffield, S10 2TN, England.

### Abstract
A method is described for generating plan-like, reflexive, obstacle avoidance behaviour in a mobile robot. The experiments reported here use a simulated vehicle with a primitive range sensor. Avoidance behaviour is encoded as a set of continuous functions of the perceptual input space. These functions are stored using CMACs and trained by a variant of Barto and Sutton's adaptive critic algorithm. As the vehicle explores its surroundings it adapts its responses to sensory stimuli so as to minimise the negative reinforcement arising from collisions. Strategies for local navigation are therefore acquired in an explicitly goal-driven fashion. The resulting trajectories form elegant collision-free paths through the environment

## 1 INTRODUCTION

Following Simon's (1969) observation that complex behaviour may simply be the reflection of a complex environment a number of researchers (eg. Braitenberg 1986, Anderson and Donath 1988, Chapman and Agre 1987) have taken the view that interesting, plan-like behaviour can emerge from the interplay of a set of pre-wired reflexes with regularities in the world. However, the temporal structure in an agent's interaction with its environment can act as more than just a trigger for fixed reactions. Given a suitable learning mechanism it can also be exploited to generate sequences of new responses more suited to the problem in hand. Hence, this paper attempts to show that obstacle avoidance, a basic level of navigation competence, can be developed through learning a set of *conditioned* responses to perceptual stimuli.

In the absence of a teacher a mobile robot can evaluate its performance only in terms of final outcomes. A negative reinforcement signal can be generated each time a collision occurs but this information tells the robot neither when nor how, in the train of actions preceding the crash, a mistake was made. In reinforcement learning this credit assignment

problem is overcome by forming associations between sensory input patterns and predictions of future outcomes. This allows the generation of internal "secondary reinforcement" signals that can be used to select improved responses. Several authors have discussed the use of reinforcement learning for navigation, this research is inspired primarily by that of Barto, Sutton and co-workers (1981, 1982, 1983, 1989) and Werbos (1990). The principles underlying reinforcement learning have recently been given a firm mathematical basis by Watkins (1989) who has shown that these algorithms are implementing an on-line, incremental, approximation to the dynamic programming method for determining optimal control. Sutton (1990) has also made use of these ideas in formulating a novel theory of classical conditioning in animal learning.

We aim to develop a reinforcement learning system that will allow a simple mobile robot with minimal sensory apparatus to move at speed around an indoor environment avoiding collisions with stationary or slow moving obstacles. This paper reports preliminary results obtained using a simulation of such a robot.

## 2   THE ROBOT SIMULATION

Our simulation models a three-wheeled mobile vehicle, called the 'sprite', operating in a simple two-dimensional world (500x500 cm) consisting of walls and obstacles in which the sprite is represented by a square box (30x30 cm). Restrictions on the acceleration and the braking response of the vehicle model enforce a degree of realism in its ability to initiate fast avoidance behaviour. The perceptual system simulates a laser range-finder giving the logarithmically scaled distance to the nearest obstacle at set angles from its current orientation. An important feature of the research has been to explore the extent to which spatially sparse but frequent data can support complex behaviour. We show below results from simulations using only three rays emitted at angles -60°, 0°, and +60°. The controller operates directly on this unprocessed sensory input. The continuous trajectory of the vehicle is approximated by a sequence of discrete time steps. In each interval the sprite acquires new perceptual data then performs the associated response generating either a change in position or a feedback signal indicating that a collision has occured preventing the move. After a collision the sprite reverses slightly then attempts to rotate and move off at a random angle (90-180° from its original heading), if this is not possible it is relocated to a random starting position.

## 3   LEARNING ALGORITHM

The sprite learns a multi-parameter policy ($\Pi$) and an evaluation (V). These functions are stored using the CMAC coarse-coding architecture (Albus 1971), and updated by a reinforcement learning algorithm similar to that described by Watkins (1989). The action functions comprising the policy are acquired as gaussian probability distributions using the method proposed by Williams (1988). The following gives a brief summary of the algorithm used.

Let $x_t$ be the perceptual input pattern at time t and $r_t$ the external reward, then the reinforcement learning error (see Barto et al., 1989) is given by

$$\varepsilon_{t+1} = r_{t+1} + \gamma V_t(x_{t+1}) - V_t(x_t) \tag{1}$$

where $\gamma$ is a constant ($0 < \gamma < 1$). This error is used to adjust V and $\Pi$ by gradient descent ie.

$$V_{t+1}(x) = V_t(x) + \alpha\, \varepsilon_{t+1}\, m_t(x) \quad \text{and} \tag{2}$$

$$\Pi_{t+1}(x) = \Pi_t(x) + \beta\, \varepsilon_{t+1}\, n_t(x) \tag{3}$$

where $\alpha$ and $\beta$ are learning rates and $m_t(x)$ and $n_t(x)$ are the evaluation and policy *eligibility traces* for pattern x. The eligibility traces can be thought of as activity in short-term memory that enables learning in the LTM store. The minimum STM requirement is to remember the last input pattern and the exploration gradient $\Delta a_t$ of the last action taken (explained below), hence

$$m_{t+1}(x) = 1 \text{ and } n_{t+1}(x) = \Delta a_t \text{ iff x is the current pattern,}$$
$$m_{t+1}(x) = n_{t+1}(x) = 0 \text{ otherwise.} \tag{4}$$

Learning occurs faster, however, if the memory trace of each pattern is allowed to decay slowly over time with strength of activity being related to recency. Hence, if the rate of decay is given by $\lambda$ ($0 <= \lambda <= 1$) then for patterns other than the current one

$$m_{t+1}(x) = \lambda\, m_t(x) \text{ and } n_{t+1}(x) = \lambda\, n_t(x).$$

Using a decay rate of less than 1.0 the eligibility trace for any input becomes negligible within a short time, so in practice it is only necessary to store a list of the most recent patterns and actions (in our simulations only the last four values are stored).

The policy acquired by the learning system has two elements (f and $\vartheta$) corresponding to the desired forward and angular velocities of the vehicle. Each element is specified by a gaussian pdf and is encoded by two adjustable parameters denoting its mean and standard deviation (hence the policy as a whole consists of four continuous functions of the input). In each time-step an action is chosen by selecting randomly from the two distributions associated with the current input pattern.

In order to update the policy the exploratory component of the action must be computed, this consists of a four-vector with two values for each gaussian element. Following Williams we define a standard gaussian density function g with parameters $\mu$ and $\sigma$ and output y such that

$$g(y,\mu,\sigma) = \frac{1}{2\sqrt{\pi}\sigma} e^{-\frac{(y-\mu)^2}{2\sigma^2}}$$

the derivatives of the mean and standard deviation[1] are then given by

$$\Delta\mu = \frac{y-\mu}{\sigma^2} \quad \text{and} \quad \Delta\sigma = \frac{[(y-\mu)^2 - \sigma^2]}{\sigma^3} \tag{5}$$

The exploration gradient of the action as a whole is therefore the vector

$$\Delta a_t = [\Delta\mu_f, \Delta\sigma_f, \Delta\mu_\vartheta, \Delta\sigma_\vartheta]. \tag{6}$$

The four policy functions and the evaluation function are each stored using a CMAC table. This technique is a form of coarse-coding whereby the euclidean space in which a function lies is divided into a set of overlapping but offset tilings. Each tiling consists of regular regions of pre-defined size such that all points within each region are mapped to a single stored parameter. The value of the function at any point is given by the average of the parameters stored for the corresponding regions in all of the tilings. In our

simulation each sensory dimension is quantised into five discrete bins resulting in a 5X5X5 tiling, five tilings are overlaid to form each CMAC. If the input space is enlarged (perhaps by adding further sensors) the storage requirements can be reduced by using a hashing function to map all the tiles onto a smaller number of parameters. This is a useful economy when there are large areas of the state space that are visited rarely or not at all.

## 4   EXPLORATION

In order for the sprite to learn useful obstacle avoidance behaviour it has to move around and explore its environment. If the sprite is rewarded simply for avoiding collisions an optimal strategy would be to remain still or to stay within a small, safe, circular orbit. Therefore to force the sprite to explore its world a second source of reinforcement is used which is a function of its current forward velocity and encourages it to maintain an optimal speed. To further promote adventurous behaviour the initial policy over the whole state-space is for the sprite to have a positive speed. A system which has a high initial expectation of future rewards will settle less rapidly for a locally optimal solution than a one with a low expectation. Therefore the value function is set initially to the maximum reward attainable by the sprite.

Improved policies are found by deviating from the currently preferred set of actions. However, there is a trade-off to be made between exploiting the existing policy to maximise the short term reward and experimenting with untried actions that have potentially negative consequences but may eventually lead to a better policy. This suggests that an annealing process should be applied to the degree of noise in the policy. In fact, the algorithm described above results in an automatic annealing process (Williams 88) since the variance of each gaussian element decreases as the mean behaviour converges to a local maximum. However, the width of each gaussian can also increase, if the mean is locally sub-optimal, allowing for more exploratory behaviour. The final width of the gaussian depends on whether the local peak in the action function is narrow or flat on top. The behaviour acquired by the system is therefore more than a set of simple reflexes. Rather, for each circumstance, there is a range of acceptable actions which is narrow if the robot is in a tight corner, where its behaviour is severely constrained, but wider in more open spaces.

## 5   RESULTS

To test the effectiveness of the learning algorithm the performance of the sprite was compared before and after fifty-thousand training steps on a number of simple environments. Over 10 independent runs[2] in the first environment shown in figure one the average distance travelled between collisions rose from approximately 0.9m (1b) before learning to 47.4m (1c) after training. At the same time the average velocity more than doubled to just below the optimal speed. The requirement of maintaining an optimum speed encourages the sprite to follow trajectories that avoid slowing down, stopping or reversing. However, if the sprite is placed too close to an obstacle to turn away safely, it can perform an n-point-turn manoeuvre requiring it to stop, back-off, turn and then move forward. It is thus capable of generating quite complex sequences of actions.

[2]Each measure was calculated over a sequence of five thousand simulation-steps with learning disabled.

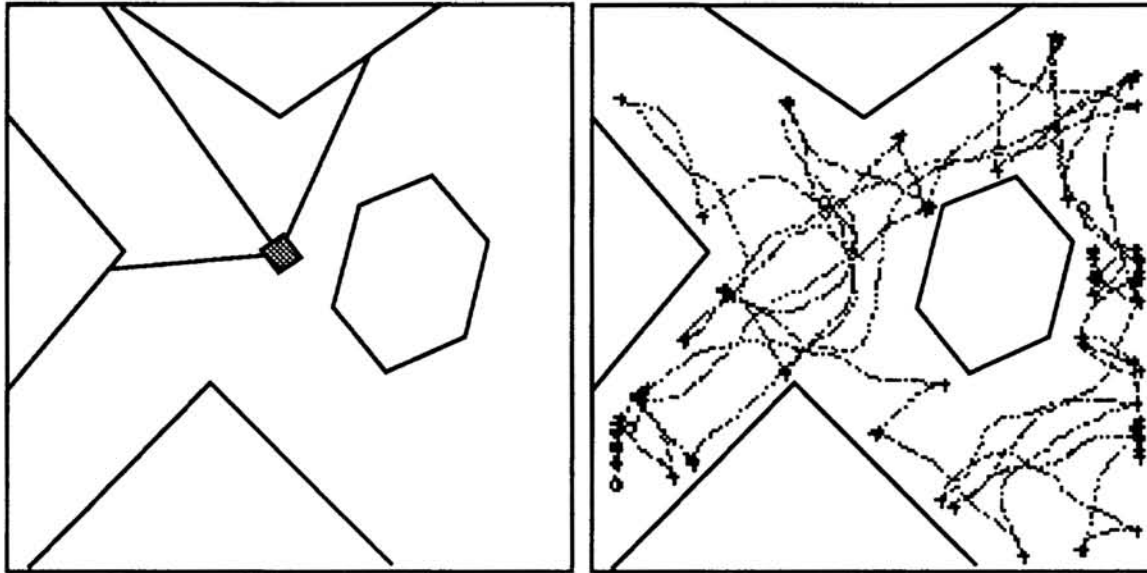

a) Robot casting three rays.          b) Trajectories before training ...

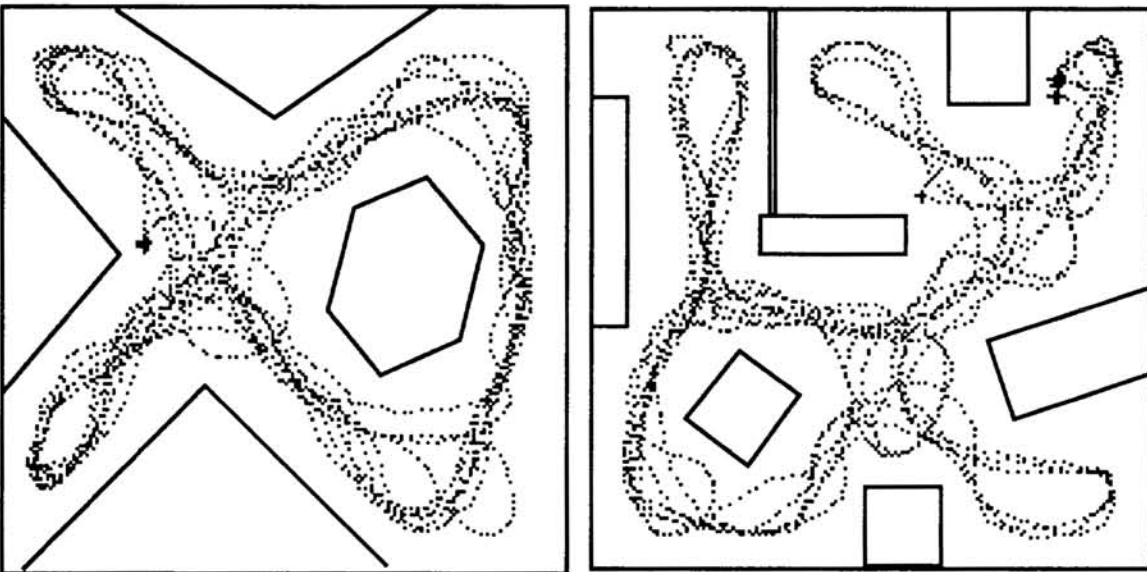

c) ... after training...              d) ... and in a novel environment

**Figure One**: Sample Paths from the Obstacle Avoidance Simulation.

The trajectories show the robot's movement over two thousand simulation steps before and after training.   After a collision the robot reverses slightly  then rotates to move off at a random angle 90-180° from its original heading, if this is not possible it is relocated to a random position.  Crosses indicate locations where collisions occured,  circles show new starting positions.

Some differences have been found in the sprite's ability to negotiate different environments with the effectiveness of the avoidance learning system varying for different configurations of obstacles. However, only limited performance loss has been observed in transferring from a learned environment to an unseen one (eg. figure 1d), which is quickly made up if the sprite is allowed to adapt its strategies to suit the new circumstances. Hence we are encouraged to think that the learning system is capturing some fairly general strategies for obstacle avoidance.

The different kinds of tactical behaviour acquired by the sprite can be illustrated using three dimensional slices through the two policy functions (desired forward and angular velocities). Figure two shows samples of these functions recorded after fifty thousand training steps in an environment containing two slow moving rectangular obstacles. Each graph is a function of the three rays cast out by the sprite: the x and y axes show the depths of the left and right rays and the vertical slices correspond to different depths of the central ray (9, 35 and 74cm). The graphs show clearly several features that we might expect of effective avoidance behaviour. Most notably, there is a transition occuring over the three slices during which the policy changes from one of braking then reversing (graph a) to one of turning sharply (d) whilst maintaining speed or accelerating (e). This transition clearly corresponds to the threshold below which a collision cannot be avoided by swerving but requires backing-off instead. There is a considerable degree of left-right symmetry (reflection along the line left-ray=right-ray) in most of the graphs. This agrees with the observation that obstacle avoidance is by and large a symmetric problem. However some asymmetric behaviour is acquired in order to break the deadlock that arises when the sprite is faced with obstacles that are equidistant on both sides.

## 6   CONCLUSION

We have demonstrated that complex obstacle avoidance behaviour can arise from sequences of learned reactions to immediate perceptual stimuli. The trajectories generated often have the appearance of planned activity since individual actions are only appropriate as part of extended patterns of movement. However, planning only occurs as an implicit part of a learning process that allows experience of rewarding outcomes to be propagated backwards to influence future actions taken in similar contexts. This learning process is effective because it is able to exploit the underlying regularities in the robot's interaction with its world to find behaviours that consistently achieve its goals.

**Acknowledgements**
This work was supported by the Science and Engineering Research Council.

**References**
Albus, J.S., (1971) A theory of cerebellar function. *Math Biosci* 10:25-61.
Anderson, T.L., and Donath, M. (1988a) Synthesis of reflexive behaviour for a mobile robot based upon a stimulus-response paradigm. *SPIE Mobile Robots III*, 1007:198-210.
Anderson, T.L., and Donath, M. (1988b) A computational structure for enforcing reactive behaviour in a mobile robot. *SPIE Mobile Robots III* 1007:370-382.
Barto, A.G., Sutton, R.S., and Brouwer, P.S. (1981) Associative search network: A reinforcement learning associative memory". *Biological Cybernetics* 40:201-211.

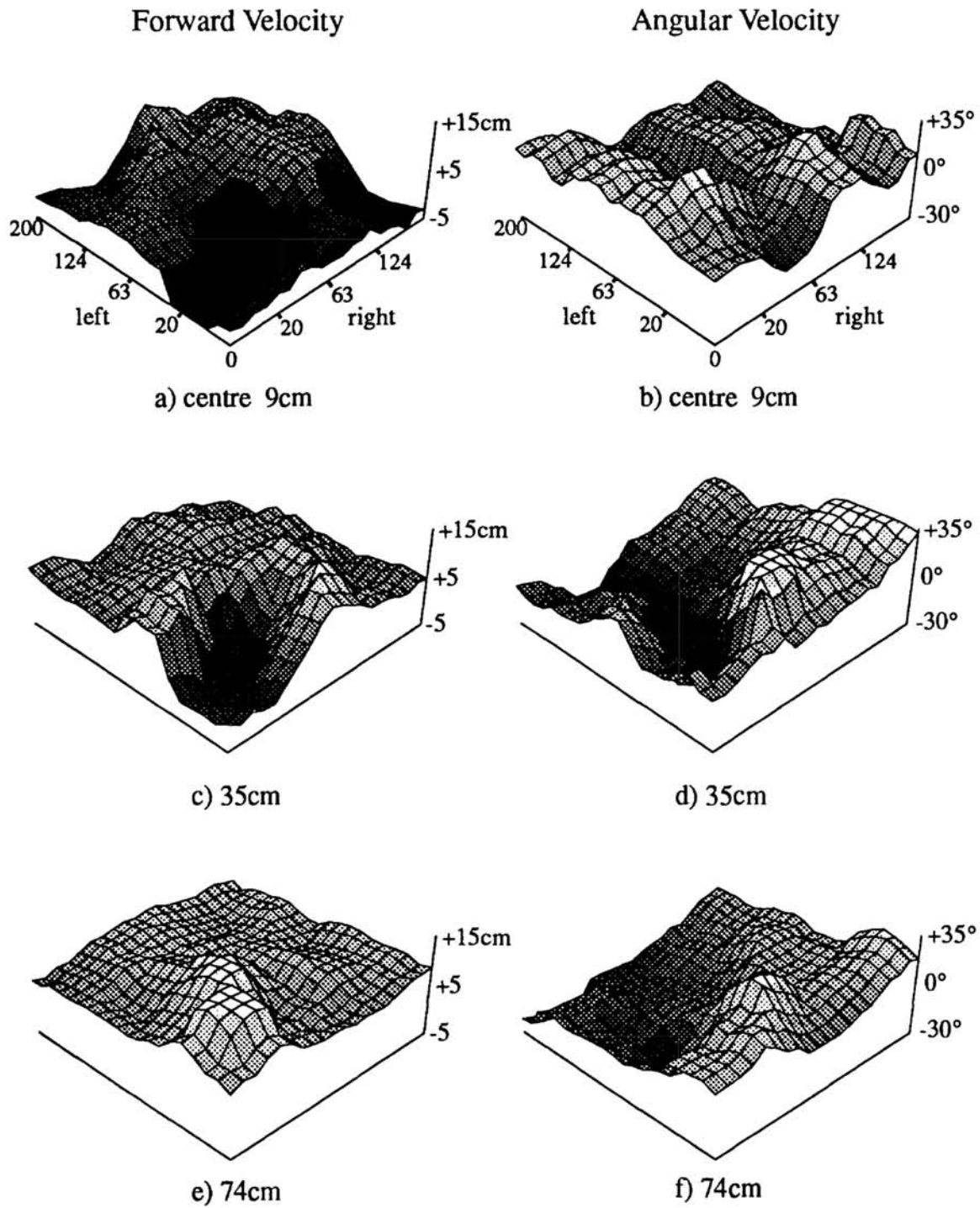

**Figure Two:** Surfaces showing action policies for depth measures
for the central ray of 9, 35 and 74 cm.

Barto, A.G., Anderson, C.W., and Sutton, R.S.(1982) Synthesis of nonlinear control surfaces by a layered associative search network. *Biological Cybernetics* 43:175-185.

Barto, A.G., Sutton, R.S., Anderson, C.W. (1983) Neuronlike adaptive elements that can solve difficult learning control problems. *IEEE Transactions on Systems, Man, and Cybernbetics* SMC-13:834-846.

Barto, A.G., Sutton, R.S., and Watkins, C.J.H.C (1989) Learning and sequential decision making. *COINS technical report.*

Braitenberg, V (1986) *Vehicles: experiments in synthetic psychology*, MIT Press, Cambridge, MA.

Chapman, D. and Agre, P.E. (1987) Pengi: An implementation of a theory of activity. AAAI-87.

Simon, H.A. (1969) *The sciences of the artificial.* MIT Press, Cambridge, Massachusetts.

Sutton, R.S. and Barto, A.G. (1990) Time-deriviative models of pavlovian reinforcement. in Moore, J.W., and Gabriel, M. (eds.) *Learning and Computational Neuroscience*, MIT Press, Cambridge, MA.

Watkins, C.J.H.C (1989) *Learning from delayed rewards.* Phd thesis, King's College, Cambridge University, UK.

Werbos, P.J. (1990) A menu of designs for reinforcement learning over time. in Millet, III, W.T., Sutton, R.S. and Werbos, P.J. *Neural networks for control*, MIT Press, Cambridge, MA.

Williams R.J., (1988) Towards a theory of reinforcement-learning connectionist systems. Technical Report NU-CCS-88-3, College of Computer Science, Northeastern University, Boston, MA.

## Footnotes

[1]In practice we use (ln s) as the second adjustable parameter to ensure that the standard deviation of the gaussian never has a negative value (see Williams 1988 for details).
